# Face Recognition Using Kernel Methods

**Ming-Hsuan Yang**
Honda Fundamental Research Labs
Mountain View, CA 94041
myang@hra.com

## Abstract

Principal Component Analysis and Fisher Linear Discriminant methods have demonstrated their success in face detection, recognition, and tracking. The representation in these subspace methods is based on second order statistics of the image set, and does not address higher order statistical dependencies such as the relationships among three or more pixels. Recently Higher Order Statistics and Independent Component Analysis (ICA) have been used as informative low dimensional representations for visual recognition. In this paper, we investigate the use of Kernel Principal Component Analysis and Kernel Fisher Linear Discriminant for learning low dimensional representations for face recognition, which we call Kernel Eigenface and Kernel Fisherface methods. While Eigenface and Fisherface methods aim to find projection directions based on the second order correlation of samples, Kernel Eigenface and Kernel Fisherface methods provide generalizations which take higher order correlations into account. We compare the performance of kernel methods with Eigenface, Fisherface and ICA-based methods for face recognition with variation in pose, scale, lighting and expression. Experimental results show that kernel methods provide better representations and achieve lower error rates for face recognition.

## 1  Motivation and Approach

Subspace methods have been applied successfully in numerous visual recognition tasks such as face localization, face recognition, 3D object recognition, and tracking. In particular, Principal Component Analysis (PCA) [20] [13],and Fisher Linear Discriminant (FLD) methods [6] have been applied to face recognition with impressive results. While PCA aims to extract a subspace in which the variance is maximized (or the reconstruction error is minimized), some unwanted variations (due to lighting, facial expressions, viewing points, etc.) may be retained (See [8] for examples). It has been observed that in face recognition the variations between the images of the same face due to illumination and viewing direction are almost always larger than image variations due to the changes in face identity [1]. Therefore, while the PCA projections are optimal in a correlation sense (or for reconstruction from a low dimensional subspace), these eigenvectors or bases may be suboptimal from the

classification viewpoint.

Representations of Eigenface [20] (based on PCA) and Fisherface [6] (based on FLD) methods encode the pattern information based on the second order dependencies, i.e., pixelwise covariance among the pixels, and are insensitive to the dependencies among multiple (more than two) pixels in the samples. Higher order dependencies in an image include nonlinear relations among the pixel intensity values, such as the relationships among three or more pixels in an edge or a curve, which can capture important information for recognition. Several researchers have conjectured that higher order statistics may be crucial to better represent complex patterns. Recently, Higher Order Statistics (HOS) have been applied to visual learning problems. Rajagopalan et al. use HOS of the images of a target object to get a better approximation of an unknown distribution. Experiments on face detection [16] and vehicle detection [15] show comparable, if no better, results than other PCA-based methods.

The concept of Independent Component Analysis (ICA) maximizes the degree of statistical independence of output variables using contrast functions such as Kullback-Leibler divergence, negentropy, and cumulants [9] [10]. A neural network algorithm to carry out ICA was proposed by Bell and Sejnowski [7], and was applied to face recognition [3]. Although the idea of computing higher order moments in the ICA-based face recognition method is attractive, the assumption that the face images comprise of a set of independent basis images (or factorial codes) is not intuitively clear. In [3] Bartlett et al. showed that ICA representation outperform PCA representation in face recognition using a subset of frontal FERET face images. However, Moghaddam recently showed that ICA representation does not provide significant advantage over PCA [12]. The experimental results suggest that seeking non-Gaussian and independent components may not necessarily yield better representation for face recognition.

In [18], Schölkopf et al. extended the conventional PCA to Kernel Principal Component Analysis (KPCA). Empirical results on digit recognition using MNIST data set and object recognition using a database of rendered chair images showed that Kernel PCA is able to extract nonlinear features and thus provided better recognition results. Recently Baudat and Anouar, Roth and Steinhage, and Mika et al. applied kernel tricks to FLD and proposed Kernel Fisher Linear Discriminant (KFLD) method [11] [17] [5]. Their experiments showed that KFLD is able to extract the most discriminant features in the feature space, which is equivalent to extracting the most discriminant nonlinear features in the original input space.

In this paper we seek a method that not only extracts higher order statistics of samples as features, but also maximizes the class separation when we project these features to a lower dimensional space for efficient recognition. Since much of the important information may be contained in the high order dependences among the pixels of a face image, we investigate the use of Kernel PCA and Kernel FLD for face recognition, which we call Kernel Eigenface and Kernel Fisherface methods, and compare their performance against the standard Eigenface, Fisherface and ICA methods. In the meanwhile, we explain why kernel methods are suitable for visual recognition tasks such as face recognition.

## 2   Kernel Principal Component Analysis

Given a set of $m$ centered (zero mean, unit variance) samples $\mathbf{x}_k$, $\mathbf{x}_k = [x_{k1}, \ldots, x_{kn}]^T \in R^n$, PCA aims to find the projection directions that maximize the variance, $C$, which is equivalent to finding the eigenvalues from the covariance

matrix

$$\lambda\mathbf{w} = C\mathbf{w} \tag{1}$$

for eigenvalues $\lambda \geq 0$ and eigenvectors $\mathbf{w} \in R^n$. In Kernel PCA, each vector $\mathbf{x}$ is projected from the input space, $R^n$, to a high dimensional feature space, $R^f$, by a nonlinear mapping function: $\Phi : R^n \to R^f, \quad f \gg n$. Note that the dimensionality of the feature space can be arbitrarily large. In $R^f$, the corresponding eigenvalue problem is

$$\lambda\mathbf{w}^\Phi = C^\Phi\mathbf{w}^\Phi \tag{2}$$

where $C^\Phi$ is a covariance matrix. All solutions $\mathbf{w}^\Phi$ with $\lambda \neq 0$ lie in the span of $\Phi(\mathbf{x}_1), \ldots, \Phi(\mathbf{x}_m)$, and there exist coefficients $\alpha_i$ such that

$$\mathbf{w}^\Phi = \sum_{i=1}^{m} \alpha_i \Phi(\mathbf{x}_i) \tag{3}$$

Denoting an $m \times m$ matrix $K$ by

$$K_{ij} = k(\mathbf{x}_i, \mathbf{x}_j) = \Phi(\mathbf{x}_i) \cdot \Phi(\mathbf{x}_j) \tag{4}$$

, the Kernel PCA problem becomes

$$m\lambda K\alpha = K^2\alpha \tag{5}$$

$$m\lambda\alpha = K\alpha \tag{6}$$

where $\alpha$ denotes a column vector with entries $\alpha_1, \ldots, \alpha_m$. The above derivations assume that all the projected samples $\Phi(\mathbf{x})$ are centered in $R^f$. See [18] for a method to center the vectors $\Phi(\mathbf{x})$ in $R^f$.

Note that conventional PCA is a special case of Kernel PCA with polynomial kernel of first order. In other words, Kernel PCA is a generalization of conventional PCA since different kernels can be utilized for different nonlinear projections.

We can now project the vectors in $R^f$ to a lower dimensional space spanned by the eigenvectors $\mathbf{w}^\Phi$, Let $\mathbf{x}$ be a test sample whose projection is $\Phi(\mathbf{x})$ in $R^f$, then the projection of $\Phi(\mathbf{x})$ onto the eigenvectors $\mathbf{w}^\Phi$ is the nonlinear principal components corresponding to $\Phi$:

$$\mathbf{w}^\Phi \cdot \Phi(\mathbf{x}) = \sum_{i=1}^{m} \alpha_i(\Phi(\mathbf{x}_i) \cdot \Phi(\mathbf{x})) = \sum_{i=1}^{m} \alpha_i k(\mathbf{x}_i, \mathbf{x}) \tag{7}$$

In other words, we can extract the first $q$ $(1 \leq q \leq m)$ nonlinear principal components (i.e., eigenvectors $\mathbf{w}^\Phi$) using the kernel function without the expensive operation that explicitly projects the samples to a high dimensional space $R^f$. The first $q$ components correspond to the first $q$ non-increasing eigenvalues of (6). For face recognition where each $\mathbf{x}$ encodes a face image, we call the extracted nonlinear principal components Kernel Eigenfaces.

## 3   Kernel Fisher Linear Discriminant

Similar to the derivations in Kernel PCA, we assume the projected samples $\Phi(\mathbf{x})$ are centered in $R^f$ (See [18] for a method to center the vectors $\Phi(\mathbf{x})$ in $R^f$), we formulate the equations in a way that use dot products for FLD only. Denoting the within-class and between-class scatter matrices by $S_W^\Phi$ and $S_B^\Phi$, and applying FLD in kernel space, we need to find eigenvalues $\lambda$ and eigenvectors $\mathbf{w}^\Phi$ of

$$\lambda S_W^\Phi \mathbf{w}^\Phi = S_B^\Phi \mathbf{w}^\Phi \tag{8}$$

, which can be obtained by

$$W_{OPT}^{\Phi} = \arg\max_{W^{\Phi}} \frac{|(W^{\Phi})^T S_B^{\Phi} W^{\Phi}|}{|(W^{\Phi})^T S_W^{\Phi} W^{\Phi}|} = [\mathbf{w}_1^{\Phi} \ \mathbf{w}_2^{\Phi} \ \dots \ \mathbf{w}_m^{\Phi}] \qquad (9)$$

where $\{\mathbf{w}_i^{\Phi}|i = 1, 2, \dots, m\}$ is the set of generalized eigenvectors corresponding to the $m$ largest generalized eigenvalues $\{\lambda_i|i = 1, 2, \dots, m\}$.

For given classes $t$ and $u$ and their samples, we define the kernel function by

$$(k_{rs})_{tu} = k(\mathbf{x}_{tr}, \mathbf{x}_{us}) = \Phi(\mathbf{x}_{tr}) \cdot \Phi(\mathbf{x}_{us}) = \Phi(\mathbf{x}_{tr})^T \Phi(\mathbf{x}_{us}) \qquad (10)$$

Let $K$ be a $m \times m$ matrix defined by the elements $(K_{tu})_{u=1,\dots,c}^{t=1,\dots,c}$, where $K_{tu}$ is a matrix composed of dot products in the feature space $R^f$, i.e.,

$$K = (K_{tu})_{u=1,\dots,c}^{=1,\dots,c} \text{ where } K_{tu} = (k_{rs})_{s=1,\dots,l_u}^{r=1,\dots,l_t} \qquad (11)$$

Note $K_{tu}$ is a $l_t \times l_u$ matrix, and $K$ is a $m \times m$ symmetric matrix. We also define a matrix $Z$:

$$Z = (Z_t)_{t=1,\dots,c} \qquad (12)$$

where $(Z_t)$ is a $l_t \times l_t$ matrix with terms all equal to $\frac{1}{l_t}$, i.e., $Z$ is a $m \times m$ block diagonal matrix. The between-class and within-class scatter matrices in a high dimensional feature space $R^f$ are defined as

$$S_B^{\Phi} = \sum_{i=1}^{c} l_i \boldsymbol{\mu}_i^{\Phi} (\boldsymbol{\mu}_i^{\Phi})^T \qquad (13)$$

$$S_W^{\Phi} = \sum_{i=1}^{c} \sum_{j=1}^{l_i} \Phi(\mathbf{x}_{ij}) \Phi(\mathbf{x}_{ij})^T \qquad (14)$$

where $\boldsymbol{\mu}_i^{\Phi}$ is the mean of class $i$ in $R^f$, $l_i$ is the number of samples belonging to class $i$. From the theory of reproducing kernels, any solution $\mathbf{w}^{\Phi} \in R^f$ must lie in the span of all training samples in $R^f$, i.e.,

$$\mathbf{w}^{\Phi} = \sum_{p=1}^{c} \sum_{q=1}^{l_p} \alpha_{pq} \Phi(\mathbf{x}_{pq}) \qquad (15)$$

It follows that we can get the solution for (15) by solving:

$$\lambda K K \alpha = K Z K \alpha \qquad (16)$$

Consequently, we can write (9) as

$$\begin{aligned} W_{OPT}^{\Phi} &= \arg\max_{W^{\Phi}} \frac{|(W^{\Phi})^T S_B^{\Phi} W^{\Phi}|}{|(W^{\Phi})^T S_W^{\Phi} W^{\Phi}|} \\ &= \arg\max_{W^{\Phi}} \frac{|\alpha K Z K \alpha|}{|\alpha K K \alpha|} \\ &= [\mathbf{w}_1^{\Phi} \dots \mathbf{w}_m^{\Phi}] \end{aligned} \qquad (17)$$

We can project $\Phi(\mathbf{x})$ to a lower dimensional space spanned by the eigenvectors $\mathbf{w}^{\Phi}$ in a way similar to Kernel PCA (See Section 2). Adopting the same technique in the Fisherface method (which avoids singularity problems in computing $W_{OPT}^{\Phi}$) for face recognition [6], we call the extracted eigenvectors in (17) Kernel Fisherfaces.

# 4 Experiments

We test both kernel methods against standard ICA, Eigenface, and Fisherface methods using the publicly available AT&T and Yale databases. The face images in these databases have several unique characteristics. While the images in the AT&T database contain the facial contours and vary in pose as well scale, the face images in the Yale database have been cropped and aligned. The face images in the AT&T database were taken under well controlled lighting conditions whereas the images in the Yale database were acquired under varying lighting conditions. We use the first database as a baseline study and then use the second one to evaluate face recognition methods under varying lighting conditions.

## 4.1 Variation in Pose and Scale

The AT&T (formerly Olivetti) database contains 400 images of 40 subjects. To reduce computational complexity, each face image is downsampled to $23 \times 28$ pixels. We represent each image by a raster scan vector of the intensity values, and then normalize them to be zero-mean vectors. The mean and standard deviation of Kurtosis of the face images are 2.08 and 0.41, respectively (the Kurtosis of a Gaussian distribution is 3). Figure 1 shows images of two subjects. In contrast to images of the Yale database, the images include the facial contours, and variation in pose as well as scale. However, the lighting conditions remain constant.

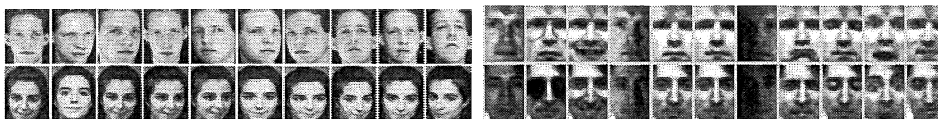

Figure 1: Face images in the AT&T database (Left) and the Yale database (Right).

The experiments are performed using the "leave-one-out" strategy: To classify an image of person, that image is removed from the training set of $(m-1)$ images and the projection matrix is computed. All the $m$ images in the training set are projected to a reduced space using the computed projection matrix $\mathbf{w}$ or $\mathbf{w}^\Phi$ and recognition is performed based on a nearest neighbor classifier. The number of principal components or independent components are empirically determined to achieve the lowest error rate by each method. Figure 2 shows the experimental results. Among all the methods, the Kernel Fisherface method with Gaussian kernel and second degree polynomial kernel achieve the lowest error rate. Furthermore, the kernel methods perform better than standard ICA, Eigenface and Fisherface methods. Though our experiments using ICA seem to contradict to the good empirical results reported in [3] [4] [2], a close look at the data sets reveals a significant difference in pose and scale variation of the face images in the AT&T database, whereas a subset of frontal FERET face images with change of expression was used in [3] [2]. Furthermore, the comparative study on classification with respect to PCA in [4] (pp. 819, Table 1) and the errors made by two ICA algorithms in [2] (pp. 50, Figure 2.18) seem to suggest that ICA methods do not have clear advantage over other approaches in recognizing faces with pose and scale variation.

## 4.2 Variation in Lighting and Expression

The Yale database contains 165 images of 11 subjects that includes variation in both facial expression and lighting. For computational efficiency, each image has been downsampled to $29 \times 41$ pixels. Likewise, each face image is represented by a

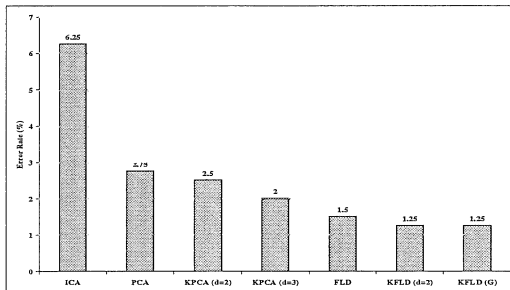

| Method | Reduced Space | Error Rate (%) |
|---|---|---|
| ICA | 40 | 6.25 (25/400) |
| Eigenface | 30 | 2.75 (11/400) |
| Fisherface | 14 | 1.50 (6/400) |
| Kernel Eigenface, d=2 | 50 | 2.50 (10/400) |
| Kernel Eigenface, d=3 | 50 | 2.00 (8/400) |
| Kernel Fisherface (P) | 14 | 1.25 (5/400) |
| Kernel Fisherface (G) | 14 | 1.25 (5/400) |

Figure 2: Experimental results on AT&T database.

centered vector of normalized intensity values. The mean and standard deviation of Kurtosis of the face images are 2.68 and 1.49, respectively. Figure 1 shows 22 closely cropped images of two subjects which include internal facial structures such as the eyebrow, eyes, nose, mouth and chin, but do not contain the facial contours.

Using the same leave-one-out strategy, we experiment with the number of principal components and independent components to achieve the lowest error rates for Eigenface and Kernel Eigenface methods. For Fisherface and Kernel Fisherface methods, we project all the samples onto a subspace spanned by the $c - 1$ largest eigenvectors. The experimental results are shown in Figure 3. Both kernel methods perform better than standard ICA, Eigenface and Fisherface methods. Notice that the improvement by the kernel methods are rather significant (more than 10%). Notice also that kernel methods consistently perform better than conventional methods for both databases. The performance achieved by the ICA method indicates that face representation using independent sources is not effective when the images are taken under varying lighting conditions.

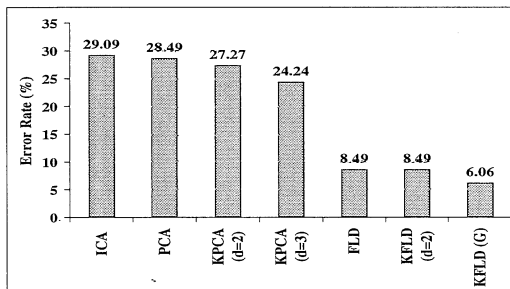

| Method | Reduced Space | Error Rate (%) |
|---|---|---|
| ICA | 30 | 29.09 (48/165) |
| Eigenface | 30 | 28.48 (47/165) |
| Fisherface | 14 | 8.48 (14/165) |
| Kernel Eigenface, d=2 | 80 | 27.27 (45/165) |
| Kernel Eigenface, d=3 | 60 | 24.24 (40/165) |
| Kernel Fisherface (P) | 14 | 6.67 (11/165) |
| Kernel Fisherface (G) | 14 | 6.06 (10/165) |

Figure 3: Experimental results on Yale database.

Figure 4 shows the training samples of the Yale database projected onto the first two eigenvectors extracted by the Kernel Eigenface and Kernel Fisherface methods. The projected samples of different classes are smeared by the Kernel Eigenface method whereas the samples projected by the Kernel Fisherface are separated quite well. In fact, the samples belonging to the same class are projected to the same position by the largest two eigenvectors. This example provides an explanation to the good results achieved by the Kernel Fisherface method.

The experimental results show that Kernel Eigenface and Fisherface methods are able to extract nonlinear features and achieve lower error rate. Instead of using a nearest neighbor classifier, the performance can potentially be improved by other classifiers (e.g., $k$-nearest neighbor and perceptron). Another potential improvement

is to use all the extracted nonlinear components as features (i.e., without projecting to a lower dimensional space) and use a linear Support Vector Machine (SVM) to construct a decision surface. Such a two-stage approach is, in spirit, similar to nonlinear SVMs in which the samples are first projected to a high dimensional feature space where a hyperplane with largest hyperplane is constructed. In fact, one important factor of the recent success in SVM applications for visual recognition is due to the use of kernel methods.

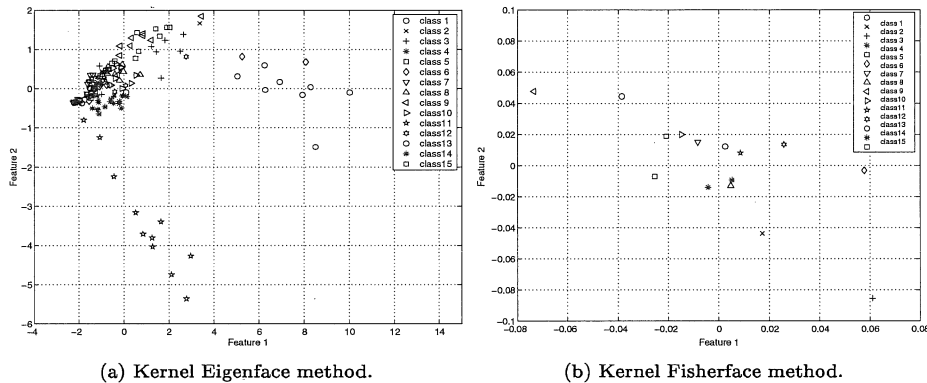

     (a) Kernel Eigenface method.          (b) Kernel Fisherface method.

Figure 4: Samples projected by Kernel PCA and Kernel Fisher methods.

## 5   Discussion and Conclusion

The representation in the conventional Eigenface and Fisherface approaches is based on second order statistics of the image set, i.e., covariance matrix, and does not use high order statistical dependencies such as the relationships among three or more pixels. For face recognition, much of the important information may be contained in the high order statistical relationships among the pixels. Using the kernel tricks that are often used in SVMs, we extend the conventional methods to kernel space where we can extract nonlinear features among three or more pixels. We have investigated Kernel Eigenface and Kernel Fisherface methods, and demonstrate that they provide a more effective representation for face recognition. Compared to other techniques for nonlinear feature extraction, kernel methods have the advantages that they do not require nonlinear optimization, but only the solution of an eigenvalue problem. Experimental results on two benchmark databases show that Kernel Eigenface and Kernel Fisherface methods achieve lower error rates than the ICA, Eigenface and Fisherface approaches in face recognition. The performance achieved by the ICA method also indicates that face representation using independent basis images is not effective when the images contain pose, scale or lighting variation. Our future work will focus on analyzing face recognition methods using other kernel methods in high dimensional space. We plan to investigate and compare the performance of other face recognition methods [14] [12] [19].

## References

[1] Y. Adini, Y. Moses, and S. Ullman. Face recognition: The problem of compensating for changes in illumination direction. *IEEE PAMI*, 19(7):721–732, 1997.

[2] M. S. Bartlett. *Face Image Analysis by Unsupervised Learning and Redundancy Reduction*. PhD thesis, University of California at San Diego, 1998.

[3] M. S. Bartlett, H. M. Lades, and T. J. Sejnowski. Independent component representations for face recognition. In *Proc. of SPIE*, volume 3299, pages 528–539, 1998.

[4] M. S. Bartlett and T. J. Sejnowski. Viewpoint invariant face recognition using independent component analysis and attractor networks. In *NIPS 9*, page 817, 1997.

[5] G. Baudat and F. Anouar. Generalized discriminant analysis using a kernel approach. *Neural Computation*, 12:2385–2404, 2000.

[6] P. Belhumeur, J. Hespanha, and D. Kriegman. Eigenfaces vs. Fisherfaces: Recognition using class specific linear projection. *IEEE PAMI*, 19(7):711–720, 1997.

[7] A. J. Bell and T. J. Sejnowski. An information - maximization approach to blind separation and blind deconvolution. *Neural Computation*, 7(6):1129–1159, 1995.

[8] C. M. Bishop. *Neural Networks for Pattern Recognition*. Oxford University Press, 1995.

[9] P. Comon. Independent component analysis: A new concept? *Signal Processing*, 36(3):287–314, 1994.

[10] A. Hyvärinen, J. Karhunen, and E. Oja. *Independent Component Analysis*. Wiley-Interscience, 2001.

[11] S. Mika, G. Rätsch, J. Weston, B. Schölkopf, A. Smola, and K.-R. Müller. Invariant feature extraction and classification in kernel spaces. In *NIPS 12*, pages 526–532, 2000.

[12] B. Moghaddam. Principal manifolds and bayesian subspaces for visual recognition. In *Proc. IEEE Int'l Conf. on Computer Vision*, pages 1131–1136, 1999.

[13] B. Moghaddam and A. Pentland. Probabilistic visual learning for object recognition. *IEEE PAMI*, 19(7):696–710, 1997.

[14] P. J. Phillips. Support vector machines applied to face recognition. In *NIPS 11*, pages 803–809, 1998.

[15] A. N. Rajagopalan, P. Burlina, and R. Chellappa. Higher order statistical learning for vehicle detection in images. In *Proc. IEEE Int'l Conf. on Computer Vision*, volume 2, pages 1204–1209, 1999.

[16] A. N. Rajagopalan, K. S. Kumar, J. Karlekar, R. Manivasakan, and M. M. Patil. Finding faces in photographs. In *Proc. IEEE Int'l Conf. on Computer Vision*, pages 640–645, 1998.

[17] V. Roth and V. Steinhage. Nonlinear discriminant analysis using kernel functions. In *NIPS 12*, pages 568–574, 2000.

[18] B. Schölkopf, A. Smola, and K.-R. Müller. Nonlinear component analysis as a kernel eigenvalue problem. *Neural Computation*, 10(5):1299–1319, 1998.

[19] Y. W. Teh and G. E. Hinton. Rate-coded restricted Boltzmann machines for face recognition. In *NIPS 13*, pages 908–914, 2001.

[20] M. Turk and A. Pentland. Eigenfaces for recognition. *J. of Cognitive Neuroscience*, 3(1):71–86, 1991.
